# Approximating Posterior Distributions in Belief Networks using Mixtures

**Christopher M. Bishop**     **Neil Lawrence**

Neural Computing Research Group
Dept. Computer Science & Applied Mathematics
Aston University
Birmingham, B4 7ET, U.K.

**Tommi Jaakkola**     **Michael I. Jordan**

Center for Biological and Computational Learning
Massachusetts Institute of Technology
79 Amherst Street, E10-243
Cambridge, MA 02139, U.S.A.

## Abstract

Exact inference in densely connected Bayesian networks is computationally intractable, and so there is considerable interest in developing effective approximation schemes. One approach which has been adopted is to bound the log likelihood using a mean-field approximating distribution. While this leads to a tractable algorithm, the mean field distribution is assumed to be factorial and hence unimodal. In this paper we demonstrate the feasibility of using a richer class of approximating distributions based on *mixtures* of mean field distributions. We derive an efficient algorithm for updating the mixture parameters and apply it to the problem of learning in sigmoid belief networks. Our results demonstrate a systematic improvement over simple mean field theory as the number of mixture components is increased.

## 1   Introduction

Bayesian belief networks can be regarded as a fully probabilistic interpretation of feed-forward neural networks. Maximum likelihood learning for Bayesian networks requires the evaluation of the likelihood function $P(V|\theta)$ where $V$ denotes the set of instantiated (visible) variables, and $\theta$ represents the set of parameters (weights and biases) in the network. Evaluation of $P(V|\theta)$ requires summing over exponentially many configurations of

the hidden variables $H$, and is computationally intractable except for networks with very sparse connectivity, such as trees. One approach is to consider a rigorous lower bound on the log likelihood, which is chosen to be computationally tractable, and to optimize the model parameters so as to maximize this bound instead.

If we introduce a distribution $Q(H)$, which we regard as an approximation to the true posterior distribution, then it is easily seen that the log likelihood is bounded below by

$$\mathcal{F}[Q] = \sum_{\{H\}} Q(H) \ln \frac{P(V,H)}{Q(H)}. \tag{1}$$

The difference between the true log likelihood and the bound given by (1) is equal to the Kullback-Leibler divergence between the true posterior distribution $P(H|V)$ and the approximation $Q(H)$. Thus the correct log likelihood is reached when $Q(H)$ exactly equals the true posterior. The aim of this approach is therefore to choose an approximating distribution which leads to computationally tractable algorithms and yet which is also flexible so as to permit a good representation of the true posterior. In practice it is convenient to consider parametrized distributions, and then to adapt the parameters to maximize the bound. This gives the best approximating distribution within the particular parametric family.

## 1.1 Mean Field Theory

Considerable simplification results if the model distribution is chosen to be factorial over the individual variables, so that $Q(H) = \prod_i Q(h_i)$, which gives *mean field theory*. Saul *et al.* (1996) have applied mean field theory to the problem of learning in sigmoid belief networks (Neal, 1992). These are Bayesian belief networks with binary variables in which the probability of a particular variable $S_i$ being on is given by

$$P(S_i = 1 | \mathrm{pa}(S_i)) = \sigma \left( \sum_j J_{ij} S_j + b_i \right) \tag{2}$$

where $\sigma(z) \equiv (1 + e^{-z})^{-1}$ is the logistic sigmoid function, $\mathrm{pa}(S_i)$ denote the parents of $S_i$ in the network, and $J_{ij}$ and $b_i$ represent the adaptive parameters (weights and biases) in the model. Here we briefly review the framework of Saul *et al.* (1996) since this forms the basis for the illustration of mixture modelling discussed in Section 3. The mean field distribution is chosen to be a product of Bernoulli distributions of the form

$$Q(H) = \prod_i \mu_i^{h_i} (1 - \mu_i)^{1-h_i} \tag{3}$$

in which we have introduced mean-field parameters $\mu_i$. Although this leads to considerable simplification of the lower bound, the expectation over the log of the sigmoid function, arising from the use of the conditional distribution (2) in the lower bound (1), remains intractable. This can be resolved by using variational methods (Jaakkola, 1997) to find a lower bound on $\mathcal{F}(Q)$, which is therefore itself a lower bound on the true log likelihood. In particular, Saul *et al.* (1996) make use of the following inequality

$$\langle \ln[1 + e^{z_i}] \rangle \leq \xi_i \langle z_i \rangle + \ln \langle e^{-\xi_i z_i} + e^{(1-\xi_i)z_i} \rangle \tag{4}$$

where $z_i$ is the argument of the sigmoid function in (2), and $\langle \ \rangle$ denotes the expectation with respect to the mean field distribution. Again, the quality of the bound can be improved by adjusting the variational parameter $\xi_i$. Finally, the derivatives of the lower bound with respect to the $J_{ij}$ and $b_i$ can be evaluated for use in learning. In summary, the algorithm involves presenting training patterns to the network, and for each pattern adapting the $\mu_i$ and $\xi_i$ to give the best approximation to the true posterior within the class of separable distributions of the form (3). The gradients of the log likelihood bound with respect to the model parameters $J_{ij}$ and $b_i$ can then be evaluated for this pattern and used to adapt the parameters by taking a step in the gradient direction.

## 2   Mixtures

Although mean field theory leads to a tractable algorithm, the assumption of a completely factorized distribution is a very strong one. In particular, such representations can only effectively model posterior distributions which are uni-modal. Since we expect multi-modal distributions to be common, we therefore seek a richer class of approximating distributions which nevertheless remain computationally tractable. One approach (Saul and Jordan, 1996) is to identify a tractable substructure within the model (for example a chain) and then to use mean field techniques to approximate the remaining interactions. This can be effective where the additional interactions are weak or are few in number, but will again prove to be restrictive for more general, densely connected networks. We therefore consider an alternative approach[1] based on mixture representations of the form

$$Q_{\text{mix}}(H) = \sum_{m=1}^{M} \alpha_m Q(H|m) \tag{5}$$

in which each of the components $Q(H|m)$ is itself given by a mean-field distribution, for example of the form (3) in the case of sigmoid belief networks. Substituting (5) into the lower bound (1) we obtain

$$\mathcal{F}[Q_{\text{mix}}] = \sum_m \alpha_m \mathcal{F}[Q(H|m)] + I(m, H) \tag{6}$$

where $I(m, H)$ is the mutual information between the component label $m$ and the set of hidden variables $H$, and is given by

$$I(m, H) = \sum_m \sum_{\{H\}} \alpha_m Q(H|m) \ln \frac{Q(H|m)}{Q_{\text{mix}}(H)}. \tag{7}$$

The first term in (6) is simply a convex combination of standard mean-field bounds and hence is no greater than the largest of these and so gives no useful improvement over a single mean-field distribution. It is the second term, i.e. the mutual information, which characterises the gain in using mixtures. Since $I(m, H) \geq 0$, the mutual information increases the value of the bound and hence improves the approximation to the true posterior.

### 2.1   Smoothing Distributions

As it stands, the mutual information itself involves a summation over the configurations of hidden variables, and so is computationally intractable. In order to be able to treat it efficiently we first introduce a set of 'smoothing' distributions $R(H|m)$, and rewrite the mutual information (7) in the form

$$
\begin{aligned}
I(m, H) \;=\; & \sum_m \sum_{\{H\}} \alpha_m Q(H|m) \ln R(H|m) - \sum_m \alpha_m \ln \alpha_m \\
& - \sum_m \sum_{\{H\}} \alpha_m Q(H|m) \ln \left\{ \frac{R(H|m)}{\alpha_m} \frac{Q_{\text{mix}}(H)}{Q(H|m)} \right\}.
\end{aligned} \tag{8}
$$

It is easily verified that (8) is equivalent to (7) for arbitrary $R(H|m)$. We next make use of the following inequality

$$-\ln x \geq -\lambda x + \ln \lambda + 1 \tag{9}$$

to replace the logarithm in the third term in (8) with a linear function (conditionally on the component label $m$). This yields a lower bound on the mutual information given by $I(m, H) \geq I_\lambda(m, H)$ where

$$I_\lambda(m, H) = \sum_m \sum_{\{H\}} \alpha_m Q(H|m) \ln R(H|m) - \sum_m \alpha_m \ln \alpha_m$$
$$- \sum_m \lambda_m \sum_{\{H\}} R(H|m) Q_{\text{mix}}(H) + \sum_m \alpha_m \ln \lambda_m + 1. \qquad (10)$$

With $I_\lambda(m, H)$ substituted for $I(m, H)$ in (6) we again obtain a rigorous lower bound on the true log likelihood given by

$$\mathcal{F}_\lambda[Q_{\text{mix}}(H)] = \sum_m \alpha_m \mathcal{F}[Q(H|m)] + I_\lambda(m, H). \qquad (11)$$

The summations over hidden configurations $\{H\}$ in (10) can be performed analytically if we assume that the smoothing distributions $R(H|m)$ factorize. In particular, we have to consider the following two summations over hidden variable configurations

$$\sum_{\{H\}} R(H|m) Q(H|k) = \prod_i \sum_{h_i} R(h_i|m) Q(h_i|k) \stackrel{\text{def}}{=} \pi_{R,Q}(m, k) \qquad (12)$$

$$\sum_{\{H\}} Q(H|m) \ln R(H|m) = \sum_i \sum_{h_i} Q(h_i|m) \ln R(h_i|m) \stackrel{\text{def}}{=} H(Q\|R|m). \qquad (13)$$

We note that the left hand sides of (12) and (13) represent sums over exponentially many hidden configurations, while on the right hand sides these have been re-expressed in terms of expressions requiring only polynomial time to evaluate by making use of the factorization of $R(H|m)$.

It should be stressed that the introduction of a factorized form for the smoothing distributions still yields an improvement over standard mean field theory. To see this, we note that if $R(H|m) = \text{const.}$ for all $\{H, m\}$ then $I(m, H) = 0$, and so optimization over $R(H|m)$ can only improve the bound.

## 2.2 Optimizing the Mixture Distribution

In order to obtain the tightest bound within the class of approximating distributions, we can maximize the bound with respect to the component mean-field distributions $Q(H|m)$, the mixing coefficients $\alpha_m$, the smoothing distributions $R(H|m)$ and the variational parameters $\lambda_m$, and we consider each of these in turn.

We will assume that the choice of a single mean field distribution leads to a tractable lower bound, so that the equations

$$\frac{\partial \mathcal{F}[Q]}{\partial Q(h_j)} = \text{const} \qquad (14)$$

can be solved efficiently[2]. Since $I_\lambda(m, H)$ in (10) is linear in the marginals $Q(h_j|m)$, it follows that its derivative with respect to $Q(h_j|m)$ is independent of $Q(h_j|m)$, although it will be a function of the other marginals, and so the optimization of (11) with respect to individual marginals again takes the form (14) and by assumption is therefore soluble.

Next we consider the optimization with respect to the mixing coefficients $\alpha_m$. Since all of the terms in (11) are linear in $\alpha_m$, except for the entropy term, we can write

$$\mathcal{F}_\lambda[Q_{\text{mix}}(H)] = \sum_m \alpha_m(-E_m) - \sum_m \alpha_m \ln \alpha_m + 1 \qquad (15)$$

where we have used (10) and defined

$$
\begin{aligned}
-E_m &= \mathcal{F}[Q(H|m)] + \sum_{\{H\}} Q(H|m) \ln R(H|m) \\
&+ \sum_k \lambda_k \sum_{\{H\}} R(H|k)Q(H|m) + \ln \lambda_m.
\end{aligned}
\tag{16}
$$

Maximizing (15) with respect to $\alpha_m$, subject to the constraints $0 \le \alpha_m \le 1$ and $\sum_m \alpha_m = 1$, we see that the mixing coefficients which maximize the lower bound are given by the Boltzmann distribution

$$
\alpha_m = \frac{\exp(-E_m)}{\sum_k \exp(-E_k)}.
\tag{17}
$$

We next maximize the bound (11) with respect to the smoothing marginals $R(h_j|m)$. Some manipulation leads to the solution

$$
R(h_j|m) = \frac{\alpha_m Q(h_j|m)}{\lambda_m} \left[ \sum_k \alpha_k \pi_{R,Q}^j(m,k) Q(h_j|k) \right]^{-1}
\tag{18}
$$

in which $\pi_{R,Q}^j(m,k)$ denotes the expression defined in (12) but with the $j$ term omitted from the product.

The optimization of the $\mu_{mj}$ takes the form of a re-estimation formula given by an extension of the result obtained for mean-field theory by Saul et al. (1996). For simplicity we omit the details here.

Finally, we optimize the bound with respect to the $\lambda_m$, to give

$$
\frac{1}{\lambda_m} = \frac{1}{\alpha_m} \sum_k \pi_{R,Q}(m,k).
\tag{19}
$$

Since the various parameters are coupled, and we have optimized them individually keeping the remainder constant, it will be necessary to maximize the lower bound iteratively until some convergence criterion is satisfied. Having done this for a particular instantiation of the visible nodes, we can then determine the gradients of the bound with respect to the parameters governing the original belief network, and use these gradients for learning.

## 3  Application to Sigmoid Belief Networks

We illustrate the mixtures formalism by considering its application to sigmoid belief networks of the form (2). The components of the mixture distribution are given by factorized Bernoulli distributions of the form (3) with parameters $\mu_{mi}$. Again we have to introduce variational parameters $\xi_{mi}$ for each component using (4). The parameters $\{\mu_{mi}, \xi_{mi}\}$ are optimized along with $\{\alpha_m, R(h_j|m), \lambda_m\}$ for each pattern in the training set.

We first investigate the extent to which the use of a mixture distribution yields an improvement in the lower bound on the log likelihood compared with standard mean field theory. To do this, we follow Saul et al. (1996) and consider layered networks having 2 units in the first layer, 4 units in the second layer and 6 units in the third layer, with full connectivity between layers. In all cases the six final-layer units are considered to be visible and have their states clamped at zero. We generate 5000 networks with parameters $\{J_{ij}, b_i\}$ chosen randomly with uniform distribution over $(-1,1)$. The number of hidden variable configurations is $2^6 = 64$ and is sufficiently small that the true log likelihood can be computed directly by summation over the hidden states. We can therefore compare the value of

the lower bound $\mathcal{F}$ with the true log likelihood $L$, using the normalized error $(L - \mathcal{F})/L$. Figure 1 shows histograms of the relative log likelihood error for various numbers of mixture components, together with the mean values taken from the histograms. These show a systematic improvement in the quality of the approximation as the number of mixture components is increased.

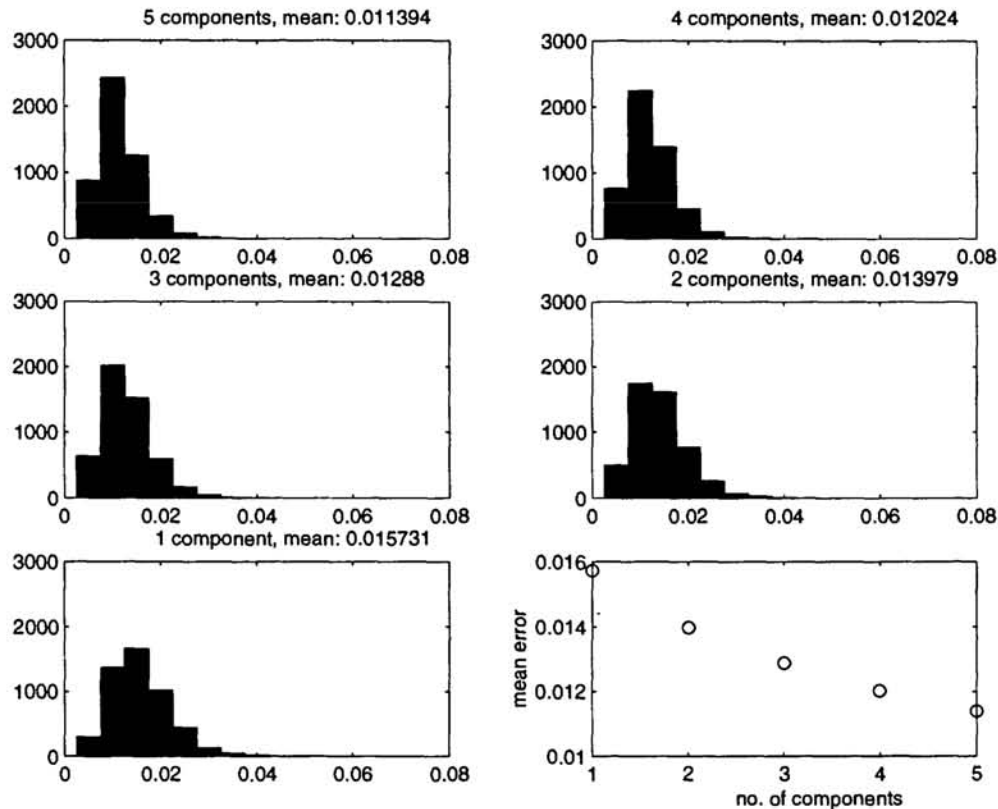

Figure 1: Plots of histograms of the normalized error between the true log likelihood and the lower bound, for various numbers of mixture components. Also shown is the mean values taken from the histograms, plotted against the number of components.

Next we consider the impact of using mixture distributions on learning. To explore this we use a small-scale problem introduced by Hinton *et al.* (1995) involving binary images of size $4 \times 4$ in which each image contains either horizontal or vertical bars with equal probability, with each of the four possible locations for a bar occupied with probability 0.5. We trained networks having architecture 1–8–16 using distributions having between 1 and 5 components. Randomly generated patterns were presented to the network for a total of 500 presentations, and the $\mu_{mi}$ and $\xi_{mi}$ were initialised from a uniform distribution over $(0, 1)$. Again the networks are sufficiently small that the exact log likelihood for the trained models can be evaluated directly. A Hinton diagram of the hidden-to-output weights for the eight units in a network trained with 5 mixture components is shown in Figure 2. Figure 3 shows a plot of the true log likelihood versus the number $M$ of components in the mixture for a set of experiments in which, for each value of $M$, the model was trained 10 times starting from different random parameter initializations. These results indicate that, as the number of mixture components is increased, the learning algorithm is able to find a set of network parameters having a larger likelihood, and hence that the improved flexibility of the approximating distribution is indeed translated into an improved training algorithm. We are currently applying the mixture formalism to the large-scale problem of hand-written digit classification.

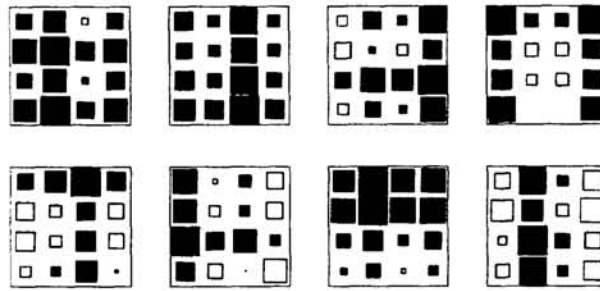

Figure 2:   Hinton diagrams of the hidden-to-output weights for each of the 8 hidden units in a network trained on the 'bars' problem using a mixture distribution having 5 components.

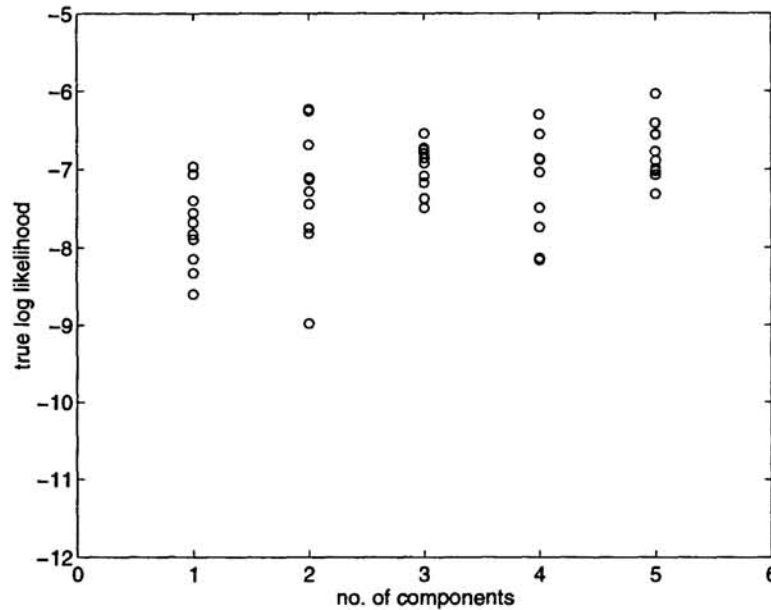

Figure 3: True log likelihood (divided by the number of patterns) versus the number $M$ of mixture components for the 'bars' problem indicating a systematic improvement in performance as $M$ is increased.

## Footnotes

[1]Here we outline the key steps. A more detailed discussion can be found in Jaakkola and Jordan (1997).

[2]In standard mean field theory the constant would be zero, but for many models of interest the slightly more general equations given by (14) will again be soluble.

## References

Hinton, G. E., P. Dayan, B. J. Frey, and R. M. Neal (1995). The wake-sleep algorithm for unsupervised neural networks. *Science* **268**, 1158–1161.

Jaakkola, T. (1997). *Variational Methods for Inference and Estimation in Graphical Models*. Ph.D. thesis, MIT.

Jaakkola, T. and M. I. Jordan (1997). Approximating posteriors via mixture models. To appear in Proceedings NATO ASI *Learning in Graphical Models*, Ed. M. I. Jordan. Kluwer.

Neal, R. (1992). Connectionist learning of belief networks. *Artificial Intelligence* **56**, 71–113.

Saul, L. K., T. Jaakkola, and M. I. Jordan (1996). Mean field theory for sigmoid belief networks. *Journal of Artificial Intelligence Research* **4**, 61–76.

Saul, L. K. and M. I. Jordan (1996). Exploiting tractable substructures in intractable networks. In D. S. Touretzky, M. C. Mozer, and M. E. Hasselmo (Eds.), *Advances in Neural Information Processing Systems*, Volume 8, pp. 486–492. MIT Press.
